# Action-Model Based Multi-agent Plan Recognition

**Hankz Hankui Zhuo**
Department of Computer Science
Sun Yat-sen University, Guangzhou, China 510006
zhuohank@mail.sysu.edu.cn

**Qiang Yang**
Huawei Noah's Ark Research Lab
Core Building 2, Hong Kong Science Park, Shatin, Hong Kong
qyang@cse.ust.hk

**Subbarao Kambhampati**
Department of Computer Science and Engineering
Arizona State University, Tempe, Arizona, US 85287-5406
rao@asu.edu

## Abstract

Multi-Agent Plan Recognition (MAPR) aims to recognize dynamic team structures and team behaviors from the observed team traces (activity sequences) of a set of intelligent agents. Previous MAPR approaches required a library of team activity sequences (team plans) be given as input. However, collecting a library of team plans to ensure adequate coverage is often difficult and costly. In this paper, we relax this constraint, so that team plans are not required to be provided beforehand. We assume instead that a set of action models are available. Such models are often already created to describe domain physics; i.e., the preconditions and effects of effects actions. We propose a novel approach for recognizing multi-agent team plans based on such action models rather than libraries of team plans. We encode the resulting MAPR problem as a *satisfiability problem* and solve the problem using a state-of-the-art weighted MAX-SAT solver. Our approach also allows for incompleteness in the observed plan traces. Our empirical studies demonstrate that our algorithm is both effective and efficient in comparison to state-of-the-art MAPR methods based on plan libraries.

## 1 Introduction

Multi-Agent Plan Recognition (MAPR) seeks an explanation of observed team-action traces. From the activity sequences of a set of agents, MAPR aims to identify the dynamic team structures and team behaviors of agents. The MAPR problem has important applications in analyzing data from automated monitoring, situation awareness, intelligence surveillance and analysis [4]. Many approaches have been proposed in the past to automatically recognize team plans given an observed team trace as input. For instance, Banerjee et al. [4, 3] proposed to formalize MAPR with a new model. They solved MAPR problems using a first-cut approach, provided that a *fully* observed team trace and a library of *full* team plans were given as input. To relax the *full observability* constraint, Zhuo and Li [19] proposed a MARS system to recognize team plans based on *partially* observed team traces and libraries of *partial* team plans.

Despite the success of these previous approaches, they all assume that a library of team plans has been collected beforehand and provided as input. However, there are many applications where collecting and maintaining a library of team plans is difficult and costly. For example, in military operations, it is difficult and expensive to collect team plans, since activities of team-mates may consume lots of resources such as ammunition and human labor. Collecting a smaller library is not an option since it is infeasible to recognize team plans if they are not covered by the library. It is thus useful to design approaches for solving the MAPR problem where we do not require libraries of team plans to be known.

In this paper, we advocate replacing the plan library with a compact action model of the domain. In contrast to plan libraries, action models are easier to specify (in terms of preconditions and effects of each type of activity). Moreover, in principle action models provide full coverage to recognize any team plans. The specific algorithmic framework we develop is called DARE, which stands for **D**omain- model based multi-**A**gent **RE**cognition, to recognize multi-agent plans. DARE does not require plan libraries to be given as input. Instead, DARE takes as input a team trace and a set of action models. DARE also allows the observed traces to be *incomplete*, i.e., there can be To fill these gaps, DARE leverages all possible constraints both from the plan traces and from its knowledge of how a plan works in terms of its causal structure. To do this, DARE first builds a set of *hard* constraints that encode the correctness property of the team plans, and a set of *soft* constraints that encode the optimal utility property of team plans based on the input team trace and action models. After that, it solves all these constraints using a state-of-the-art weighted MAX-SAT solver, such as MaxSatz [10], and converts the solution to a set of team plans as output.

We organize the rest of the paper as follows. In the next section, we first introduce the related work including single agent plan recognition and multi-agent plan recognition, and then give our formulation of the MAPR problem. After that, we present DARE and discuss its properties. Finally, we evaluate DARE in the experimental section and present our conclusions.

## 2 Related work

The plan recognition problem has been addressed by many researchers. Kautz and Allen proposed an approach to recognize plans based on parsing observed actions as sequences of subactions and essentially model this knowledge as a context-free rule in an "action grammar" [9]. Bui et al. presented approaches to probabilistic plan recognition problems [5, 7]. Instead of using a library of plans, Ramrez and Geffner [12] proposed an approach to solving the plan recognition problem using slightly modified planning algorithms, assuming the action models were given as input. Note that action models can be created by experts or learnt by previous systems, such as ARMS [18] and LAMP [20]. Singla and Mooney proposed an approach to abductive reasoning using a first-order probabilistic logic to recognize plans [15]. Amir and Gal addressed a plan recognition approach to recognizing student behaviors using virtual science laboratories [1]. Ramirez and Geffner exploited off-the-shelf classical planners to recognize probabilistic plans [13]. Despite the success of these systems, a limitation is that they all focus only on single agent plans.

For multi-agent plan recognition, Sukthankar and Sycara presented an approach that leveraged several types of agent resource dependencies and temporal ordering constraints in the plan library to prune the size of the plan library considered for each observation trace [16]. Avrahami-Zilberbrand and Kaminka preferred a library of single agent plans to team plans, but identified dynamic teams based on the assumption that all agents in a team executing the same plan under the temporal constraints of that plan [2]. The constraint on activities of the agents that can form a team can be severely limiting when team-mates can execute coordinated but different behaviors.

Instead of using the assumption that agents in the same team should execute a common activity, besides the approaches introduced in the introduction section [4, 3, 19], Sadilek and Kautz provided a unified framework to model and recognize activities that involved multiple related individuals playing a variety of roles [14]; Masato et al. proposed a probabilistic model based on conditional random fields to automatically recognize the composition of teams and team activities in relation to a plan [11]. In these systems, although coordinated activities can be recognized, they either assume there is a set of real-world GPS data available, or assume that team traces and team plans can be fully observed. In this paper, we allow that: (1) agents can execute coordinated different activities in a team, (2) team traces can be partial, and (3) neither GPS data nor team plans are needed.

# 3 Problem Definition

We first define a team trace. Let $\Phi = \{\phi_1, \phi_2, \ldots, \phi_n\}$ be a set of agents, and $\mathcal{O} = [o_{tj}]$ be an observed team trace. Let $o_{tj}$ be the observed activity executed by agent $j$ at time step $t$, where $0 < t \leq T$ and $0 < j \leq n$. A team trace $\mathcal{O}$ is *partial*, if some elements in $\mathcal{O}$ are empty (denoted by *null*), i.e., there are missing values in $\mathcal{O}$.

We then define an action model. In the STRIPS language [6], an action model is a tuple $\langle a, \mathrm{Pre}(a), \mathrm{Add}(a), \mathrm{Del}(a) \rangle$, where $a$ is an action name with zero or more parameters, $\mathrm{Pre}(a)$ is a list of preconditions of $a$, $\mathrm{Add}(a)$ is a list of add effects, and $\mathrm{Del}(a)$ is a list of deleting effects. A set of action models is denoted by $\mathcal{A}$. An action name with zero of more parameters is called an *activity*. An observed activity $o_{tj}$ in a partial team trace $\mathcal{O}$ is either an instantiated action of $\mathcal{A}$ or *noop* or *null*, where *noop* is an empty activity that does nothing.

An initial state $s_0$ is a set of propositions that describes a *closed* world state from which the team trace $\mathcal{O}$ starts to be observed. In other words, activities at time step $t = 0$ can be *applied* in the initial state $s_0$. When we say an activity can be applied in a state, we mean the activity's preconditions are satisfied by the state. A set of goals $G$, each of which is a set of propositions, describes the probable targets of the team trace. We assume $s_0$ and $G$ can both be observed by sensing devices.

A team is composed of a subset of agents $\Phi' = \{\phi_{j_1}, \phi_{j_2}, \ldots, \phi_{j_m}\}$. A team plan is defined as $p = [a_{tk}]_{0<t\leq T}^{0<k\leq m}$, where $m \leq n$, and $a_{tk}$ is an activity or *noop*. A set of *correct* team plans $P$ is required to have properties **P1**-**P5**.

**P1:** $P$ is a partition of the team trace $\mathcal{O}$, i.e., each element of $\mathcal{O}$ should be in exactly one $p$ of $P$ and each activity of $p$ should be an element of $\mathcal{O}$;

**P2:** $P$ should cover all the observed activities, i.e., for each $p \in P$ and $0 < t \leq T$ and $0 < k \leq m$, if $o_{tj_k} \neq null$, then $a_{tk} = o_{tj_k}$, where $a_{tk} \in p$ and $o_{tj_k} \in \mathcal{O}$;

**P3:** $P$ is executable starting from $s_0$ and achieves some goal $g \in G$, i.e., $a_{t*}$ is executable in state $s_{t-1}$ for all $0 < t \leq T$, and achieves $g$ after step $T$, where $a_{t*} = \langle a_{t1}, a_{t2}, \ldots, a_{tm} \rangle$;

**P4:** Each team plan $p \in P$ is associated with a likelihood $\mu(p)$: $P \mapsto R^+$. $\mu(p)$ specifies the likelihood of recognizing team plan $p$, and can be affected by many factors, including the number of agents in the team, the cost of executing $p$, etc. The value of $\mu(p)$ is composed of two parts $\mu_1(\mathcal{N}_{activity}(p))$ and $\mu_2(\mathcal{N}_{agent}(p))$, i.e., $\mu(p) = \frac{1}{\mu_1(\mathcal{N}_{activity}(p)) + \mu_2(\mathcal{N}_{agent}(p))}$, where $\mu_1(\mathcal{N}_{activity}(p))$ depends on $\mathcal{N}_{activity}(p)$, the number of activities of $p$, and $\mu_2(\mathcal{N}_{agent}(p))$ depends on $\mathcal{N}_{agent}(p)$, the number of agents (i.e., team-mates) of $p$. Generally, $\mu_1(\mathcal{N}_{activity}(p))$ (or $\mu_2(\mathcal{N}_{agent}(p))$) becomes larger when $\mathcal{N}_{activity}(p)$ (or $\mathcal{N}_{agent}(p)$) increases. Note that more agents would have a smaller likelihood (or larger cost) to coordinate these agents to successfully execute $p$. Thus, we require that $\mu_2$ should satisfy the condition: $\mu_2(n_1 + n_2) > \mu_2(n_1) + \mu_2(n_2)$. For each goal $g \in G$, the output plan $P$ should have the largest likelihood, i.e.,

$$P = \arg\max_{P'} \sum_{p \in P'} \mu(p),$$

where $P'$ is a team-plan set that achieves $g$. Note that we presume that teams are (usually) organized with the largest likelihood.

**P5:** Any pair of interacting agents must belong to the same team plan. In other words, if an agent $\phi_i$ interacts with another agent $\phi_j$, i.e., $\phi_i$ provides or deletes some conditions of $\phi_j$, then $\phi_i$ and $\phi_j$ should be in the same team, and activities of agents in the same team compose a team plan. Agents exist in *exactly* one team plan, i.e., team plans do not share any common agents.

Our multi-agent plan recognition problem can be stated as: *Given a partially observed team trace $\mathcal{O}$, a set of action models $\mathcal{A}$, an initial state $s_0$, and a set of goals $G$, the recognition algorithm must output a set of team plans $P$ with the maximal likelihood to achieve some goal $g \in G$, where $P$ satisfies the properties **P1**-**P5**.*

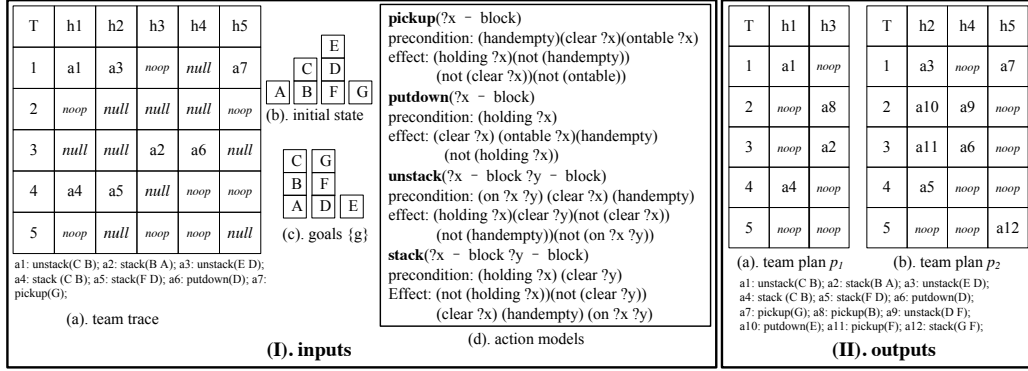

Figure 1: An example of the input and output of our problem from the *blocks* domain. (I) is an input example, where "(b) the initial state $s_0$" is a set of propositions: {(ontable A)(ontable B)(ontable F)(ontable G)(on C B)(on D F)(on E D)(clear A)(clear C)(clear E)(clear G)(handempty)}; "(c) goals {$g$}" is a goal set composed of one goal $g$, and $g$ is composed of propositions: {(ontable A)(ontable D)(ontable E)(on B A)(on C B)(on F D)(on G F)(clear C)(clear G)(clear E)}. (II) is an output example, which is the set of team plans {$p_1, p_2$}.

Figure 1 shows an example multi-agent plan recognition problem from *blocks world*[1]. In part (a) of Figure 1(I), the first column indicates the time steps from 1 to 5. $h_1, \ldots, h_5$ are five *hoist* agents. The value *null* suggests the missing observation, and *noop* suggests the empty activity. We assume $\mu_1$ and $\mu_2$ are defined by: $\mu_1(k) = k$, and $\mu_2(k) = k^2$. Based on $\mu_1$ and $\mu_2$, the corresponding output is shown in Figure 1(II), which is the set of two team plans {$p_1, p_2$}.

## 4  DARE **Algorithm Framework**

Algorithm 1 below describes the plan recognition process in DARE. In the subsequent subsections, we describe each step of this algorithm in detail.

---

**Algorithm 1** An overview of our algorithm framework

---

**input:** a partial team trace $\mathcal{O}$, an initial state $s_0$, a set of goals $G$, and a set of action models $\mathcal{A}$;
**output:** a set of team plans $P$;
 1: $max = 0$;
 2: **for** each $g \in G$ **do**
 3:     build a set of candidate activities $\Theta$;
 4:     build a set of hard constraints based on $\Theta$;
 5:     build a set of soft constraints based on the likelihood $\mu$;
 6:     solve all the constraints using a weighted MAX-SAT solver, with $\langle max', sol \rangle$ as output;
 7:     **if** $max' > max$ **then**
 8:        $max = max'$;
 9:        convert the solution *sol* to a set of team plans $P'$, and let $P = P'$;
10:     **end if**
11: **end for**
12: **return** $P$;

---

### 4.1  Candidate activities

In Step 3 of Algorithm 1, we build a set of *candidate activities* $\Theta$ by instantiating each parameter of action models in $\mathcal{A}$ with all objects in the initial state $s_0$, team trace $\mathcal{O}$ and goal $g$. We perform the following phases. We first scan each parameter of propositions (or activities) in $s_0$, $\mathcal{O}$, and $g$, and collect sets of different objects (note that each set of objects corresponds to a *type*, e.g., there

is a *type* "block" in the *blocks* domain). Second, we substitute each parameter of each action model in $\mathcal{A}$ with its corresponding objects (the correspondence relationship is reflected by *type*, i.e., the parameters of action models and objects should belong to the same *type*), which results in a set of different activities, called *candidate activities* $\Theta$. Note that we also add an *noop* activity in $\Theta$.

*For example, there are seven objects {A, B, C, D, E, F, G} corresponding to* type *"block" in Figure 1(I). The set of candidate activities $\Theta$ is: {noop, pickup(A), pickup(B), pickup(C), pickup(D), pickup(E), pickup(F), pickup(G), ... }, where the "dots" suggests other activities that are generated by instantiating parameters of actions "putdown, stack, unstack".*

## 4.2 Hard constraints

With the set of candidate activities $\Theta$, we build a set of hard constraints to ensure the properties **P1** to **P3** in Step 4 of Algorithm 1. We associate each element $o_{tj} \in \mathcal{O}$ with a *variable* $v_{tj}$, i.e., we have a set of variables $V = [v_{tj}]_{0<t\leq T}^{0<j\leq n}$, which is also called a *variable matrix*. Each variable in the variable matrix will be assigned with a specific activity in candidate activities $\Theta$, and we will partition these variables to attain a set of team plans that have the properties **P1**-**P5** based on the assignments. According to properties **P2** and **P3**, we build two kinds of hard constraints: *Observation constraints* and *Causal-link constraints*. Note that **P1** is guaranteed since the set of team plans that is output is a partition of the team trace.

**Observation constraints** For **P2**, i.e., given a team plan $p = [a_{tk}]_{0<t\leq T}^{0<k\leq m}$ composed of agents $\Phi' = \{\phi_{j_1}, \phi_{j_2}, \ldots, \phi_{j_m}\}$, if $o_{tj_k} \neq null$, then $a_{tk} = o_{tj_k}$, this suggests $v_{tj_k}$ should have the same activity of $o_{tj_k}$ if $o_{tj_k} \neq null$, since the team plan $p$ is a partition of $V$ and $a_{tk}$ is an element of of $p$. Thus, we build hard constraints as follows. For each $0 < t \leq T$ and $0 < j \leq n$, we have

$$(o_{tj} \neq null) \rightarrow (v_{tj} = o_{tj}).$$

We call this kind of hard constraints the *observation constraints*, since they are built based on the partially observed activities of $\mathcal{O}$.

**Causal-link constraints** For **P3**, i.e., each team plan $p$ should be executable starting from the initial state $s_0$, this suggests each row of variables $\langle v_{t1}, v_{t2}, \ldots, v_{tn} \rangle$ should be executable, where $0 < t \leq T$. Note that "executable" suggests that the preconditions of $v_{tj}$ should be satisfied. This means, for each $0 < t \leq T$ and $0 < j \leq n$, the following constraints should be satisfied:

- each precondition of $v_{tj}$ either exists in the initial state $s_0$ or is added by $v_{t'j'}$, and is not deleted by any activity between $t'$ and $t$, where $t' < t$ and $0 < j' \leq n$.
- likewise, each proposition in goal $g$ either exists in the initial state $s_0$ or is added by $v_{t'j'}$, and is not deleted by any activity between $t'$ and $T$, where $t' < T$ and $0 < j' \leq n$.

We call this kind of hard constraints *causal-link constraints*, since they are created according to the causal link requirement of executable plans.

## 4.3 Soft constraints

In Step 5 of Algorithm 1, we build a set of soft constraints based on the likelihood function $\mu$. Each variable in $V$ can be assigned with any element of the candidate activities $\Theta$. We require that all variables in $V$ should be assigned with exactly one activity from $\Theta$. For each $\langle a_1, a_2, \ldots, a_{|V|} \rangle \in \Theta \times \ldots \times \Theta$, we have

$$\bigwedge_{0<i\leq|V|} (v_i = a_i).$$

We calculate the weights of these constraints by the following phases. First, we partition the variable matrix $V$ based on property **P5** into a set of team plans $P$, i.e., agent $\phi_i$ provides or deletes some conditions of $\phi_j$, then $\phi_i$ and $\phi_j$ should be in the same team, and activities of agents in the same team compose a team plan. Second, for all team plans, we calculate the total likelihood $\mu(P)$, i.e.,

$$\mu(P) = \sum_{p \in P} \mu(p) = \sum_{p \in P} \frac{1}{\mu_1(\mathcal{N}_{activity}(p)) + \mu_2(\mathcal{N}_{agent}(p))},$$

and let $\mu(P)$ be the weights of the soft constraints. Note that we aim to maximize the total likelihood when solving these constraints (together with hard constraints) with a weighted MAX-SAT solver.

## 4.4 Solving the constraints

In Step 6 of Algorithm 1, we put both hard and soft constraints together, and solve these constraints using Maxsatz [10], a MAX-SAT solver . The solution $sol$ is an assignment for all variables in $V$, and $max'$ is the total weight of the satisfied constraints corresponding to the solution $sol$. In Step 8 of Algorithm 1, we partition $V$ into a set of team plans $P$ based on **P5**.

As an example, in (a) of Figure 1(I), the team trace's corresponding variable in $V$ is assigned with activities, which means the *null* values in (a) of Figure 1(I) are replaced with the corresponding assigned activities in $V$. According to property **P5**, we can simply partition the team trace into two team plans, as is shown in Figure 1(II), by checking preconditions and effects of activities in the team trace.

## 4.5 Properties of DARE

DARE can be shown to have the following properties:

**Theorem 1: (Conditional Soundness)** If the weighted MAX-SAT solver is powerful enough to optimally solve all solvable SAT problems, DARE is sound.

**Theorem 2: (Conditional Completeness)** If the weighted MAX-SAT solver we exploit in DARE is complete, DARE is also complete.

For Theorem 1, we only need to check that the solutions output by DARE satisfy **P1**-**P5**. **P2** and **P3** are guarranteed by observation constraints and causal-link constraints; **P4** is guaranteed by the soft constraints built in Section 4.3 and the MAX-SAT solver; **P1** and **P5** are both guaranteed by the partition step in Section 4.4, i.e., partitioning the variable matrix into a set of team plans; that is to say, the conditional soundness property holds.

For Theorem 2, since all steps in Algorithm 1, except Step 6 that calls a weighted MAX-SAT solver, can be executed in finite time, the completeness property only depends on the weighted MAX-SAT solver, which means the conditional completeness property holds.

# 5 Experiments

## 5.1 Dataset and Evaluation Criterion

We evaluate DARE in three planning domains: *blocks*, *driverlog*[2] and *rovers*[2]. We modify the three domains for multi-agent setting. In *blocks*, there are multiple *hoists*, which are viewed as agents that perform actions of *pickup*, *putdown*, *stack* and *unstack*. In *driverlog*, there are multiple *trucks*, *drivers* and *hoists*, which are agents that can group together to form different teams (trucks and drivers can be in the same team, likewise for hoists.). In *rovers*, there are multiple *rovers* that can group together to form different teams. For each domain, we set $T = 50$ and generate 50 team traces with the size of $T \times n$ for each $n \in \{20, 40, 60, 80, 100\}$. For each team trace, we have a set of optimal team plans (which is viewed as the *ground truth*), denoted by $P_{true}$, and its corresponding goal $g_{true}$, which best explains the team trace according to the likelihood function $\mu$. We define the likelihood function by: $\mu = \mu_1 + \mu_2$, where $\mu_1(k) = k$ and $\mu_2(k) = k^2$, as is presented in the end of the problem definition section.

We randomly delete a subset of activities from each team trace with respect to a specific percentage $\xi$. We will test different $\xi$ values with 0%, 10%, 20%, 30%, 40%, 50%. As an example, $\xi = 10\%$ suggests there are 10 activities deleted from a team trace with 100 activities. We also randomly add 10 additional goals, together with $g_{true}$, to form the goal set $G$, as is presented in the problem definition section. We define the accuracy $\lambda$ by:

$$\lambda = \frac{\text{the number of correctly recognized team plan sets}}{\text{the total number of team traces}},$$

where "correctly recognized team plan sets" suggests the recognized team plan sets and goals are the same as the expected team plan sets $\{P_{true}\}$ and goals $G$.

We generate 100 team plans as the library as is described by MARS [19], and compare the recognition results with MARS as a baseline.

## 5.2 Experimental Results

We evaluate DARE in the following aspects: (1) accuracy with respect to different number of agents; (2) accuracy with respect to different percentages of *null* values; and (3) the running time.

### 5.2.1 Varying the number of agents

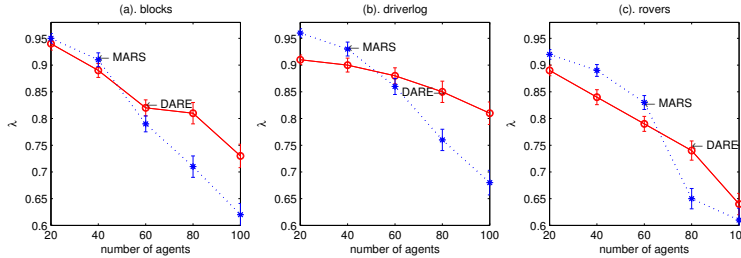

Figure 2: Accuracies with respect to different number of agents

We would like to evaluate the change of accuracies when the number of agents increases. We set the percentage of *null* values to be 30%, and also ran DARE five times to calculate an average of accuracies. The result is shown in Figure 2. From the figure, we found that the accuracies of both DARE and MARS generally decreased when the number of agents increased. This is because the problem space is enlarged when the number of agents increases, which makes the available information be decreased comparing to the large problem space, and not enough to attain high accuracies.

We also found that the accuracy of DARE was lower than MARS at the beginning, and then became better than MARS as the number of agents became larger. This indicates that DARE has better performance in handling large number of agents based on action models. This is because DARE builds the MAX-SAT problem space (described as proposition variables and constraints) based on model inferences (i.e., action models), while MARS is based on instances (i.e., plan library). When the number of agents is small, the problem space built by MARS is smaller than that built by DARE; when the number of agents becomes larger, the problem space built by MARS becomes larger than that built by DARE; the larger the problem space is, the more difficult it is for MAX-SAT to solve the problem; thus, DARE performs worse than MARS with less agents, while better with more agents.

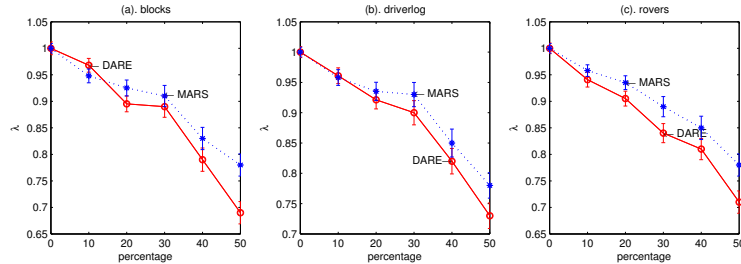

Figure 3: Accuracies with respect to different percentages of *null* values.

### 5.2.2 Varying the percentage of *null* values

We set the number of agents to be 60, and run DARE five times to calculate average of accuracies with a percentage $\xi$ of *null* values. We found both accuracies of DARE and MARS decreased when

the percentage $\xi$ increased, due to less information provided when the percentage increasing. When the percentage is 0%, both DARE and MARS can recognize all the team traces successfully.

By observing all three domains in Figure 3, we find that DARE does not function as well as MARS when the percentage of incompleteness is large. This relative advantage for the library-based approach is due in large part to the fact that all team plans to be recognized are covered by the small library in the experiment, and the library of team plans will help reduce the recognition problem space compared to DARE. We conjecture that if the team plans to be recognized are not covered by the library (because of the size restrictions on the library), DARE will perform better than MARS. In this case, MARS cannot successfully recognize some team plans.

### 5.2.3   The running time

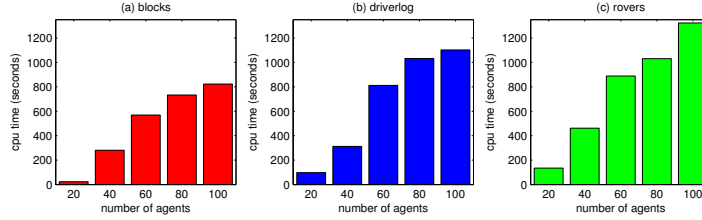

Figure 4: The CPU time of DARE.

We show the average CPU time of DARE over 50 team traces with respect to different number of agents in Figure 4. As can be seen from the figure, the running time increases polynomially with the number of input agents. This can be verified by fitting the relationship between the number of agents and the running time to a performance curve with a polynomial of order 2 or 3. For example, the fit polynomial for *blocks* is $-0.0821x^2 + 20.1171x - 359.8$.

## 6   Final Remark

In this paper, we presented a system called DARE for recognizing multi-agent team plans from incomplete observed plan traces based on action models . This approach has significant advantage over previous approaches that make use of a library of predefined team plans. Such plan libraries are difficult to obtain in many applications. With the action model based approach, we first build a set of candidate activities, and then build sets of hard and soft constraints to finally recognize team plans. Our experiments show that DARE is effective in three benchmark domains compared to the state-of-the-art multi-agent plan recognition system MARS that relies on a library of team plans. Our approach is thus well suited for scenarios where collecting a library of team plans is infeasible before performing team plan recognition tasks.

In the current work, we assume that the action models are *complete*. A more realistic assumption is to allow the models to be incomplete [8, 17]. In future, we plan to extend DARE to work with incomplete action models. Another assumption in the current model is that it expects as input the alternative sets of goals, one of which the observed plan is expected to be targeting. We plan to relax this so DARE can take as input a set of potential goals, with the understanding that the observed plan is achieving a bounded subset of these goals. We believe that both these extensions can be easily accommodated into the MAX-SAT framework of DARE.

### Acknowledgments

Hankz Hankui Zhuo thanks Natural Science Foundation of Guangdong Province of China (No. S2011040001869) and Research Fund for the Doctoral Program of Higher Education of China (No. 20110171120054) for the support of this research. Qiang Yang thanks Hong Kong RGC GRF Projects 621010 and 621211 for the support of this research. Kambhampati's research is supported in part by the NSF grant IIS201330813 and ONR grants N00014-09-1-0017, N00014-07-1-1049, and N000140610058.

## Footnotes

[1]http://www.cs.toronto.edu/aips2000/

[2]http://planning.cis.strath.ac.uk/competition/

# References

[1] Ofra Amir and Yaakov (Kobi) Gal. Plan recognition in virtual laboratories. In *Proceedings of IJCAI*, 2011.

[2] Dorit Avrahami-Zilberbrand and Gal A. Kaminka. Towards dynamic tracking of multi-agents teams: An initial report. In *Proceedings of the AAAI Workshop on Plan, Activity, and Intent Recognition (PAIR 2007)*, 2007.

[3] Bikramjit Banerjee and Landon Kraemer. Branch and price for multi-agent plan recognition. In *Proceedings of AAAI*, 2011.

[4] Bikramjit Banerjee, Landon Kraemer, and Jeremy Lyle. Multi-agent plan recognition: formalization and algorithms. In *Proceedings of AAAI*, 2010.

[5] Hung H. Bui. A general model for online probabilistic plan recognition. In *Proceedings of IJCAI*, 2003.

[6] R. Fikes and N. J. Nilsson. STRIPS: A new approach to the application of theorem proving to problem solving. *Artificial Intelligence Journal*, pages 189–208, 1971.

[7] Christopher W. Geib and Robert P. Goldman. A probabilistic plan recognition algorithm based on plan tree grammars. *Artificial Intelligence*, 173(11):1101–1132, 2009.

[8] Subbarao Kambhampati. Model-lite planning for the web age masses: The challenges of planning with incomplete and evolving domain models. In *AAAI*, 2007.

[9] Henry A. Kautz and James F. Allen. Generalized plan recognition. In *Proceedings of AAAI*, 1986.

[10] Chu Min LI, Felip Manya, Nouredine Mohamedou, and Jordi Planes. Exploiting cycle structures in Max-SAT. In *In proceedings of 12th international conference on the Theory and Applications of Satisfiability Testing (SAT-09)*, pages 467–480, 2009.

[11] Daniele Masato, Timothy J. Norman, Wamberto W. Vasconcelos, and Katia Sycara. Agent-oriented incremental team and activity recognition. In *Proceedings of IJCAI*, 2011.

[12] Miquel Ramrez and Hector Geffner. Plan recognition as planning. In *Proceedings of IJCAI*, 2009.

[13] Miquel Ramrez and Hector Geffner. Probabilistic plan recognition using off-the-shelf classical planners. In *Proceedings of AAAI*, 2010.

[14] Adam Sadilek and Henry Kautz. Recognizing multi-agent activities from gps data. In *Proceedings of AAAI*, 2010.

[15] Parag Singla and Raymond Mooney. Abductive markov logic for plan recognition. In *Proceedings of AAAI*, 2011.

[16] Gita Sukthankar and Katia Sycara. Hypothesis pruning and ranking for large plan recognition problems. In *Proceedings of AAAI*, 2008.

[17] Minh Do. Tuan Nguyen, Subbarao Kambhampati. Synthesizing robust plans under incomplete domain models. In *Proc. AAAI Workshop on Generalized Planning*, 2011.

[18] Qiang Yang, Kangheng Wu, and Yunfei Jiang. Learning action models from plan examples using weighted MAX-SAT. *Artificial Intelligence*, 171:107–143, February 2007.

[19] Hankz Hankui Zhuo and Lei Li. Multi-agent plan recognition with partial team traces and plan libraries. In *Proceedings of IJCAI*, 2011.

[20] Hankz Hankui Zhuo, Qiang Yang, Derek Hao Hu, and Lei Li. Learning complex action models with quantifiers and implications. *Artificial Intelligence*, 174(18):1540 – 1569, 2010.

